# Consistency of one-class SVM and related algorithms

**Régis Vert**
Laboratoire de Recherche en Informatique
Université Paris-Sud
91405, Orsay Cedex, France
Masagroup
24 Bd de l'Hôpital
75005, Paris, France
Regis.Vert@lri.fr

**Jean-Philippe Vert**
Geostatistics Center
Ecole des Mines de Paris - ParisTech
77300 Fontainebleau, France
Jean-Philippe.Vert@ensmp.fr

## Abstract

We determine the asymptotic limit of the function computed by support vector machines (SVM) and related algorithms that minimize a regularized empirical convex loss function in the reproducing kernel Hilbert space of the Gaussian RBF kernel, in the situation where the number of examples tends to infinity, the bandwidth of the Gaussian kernel tends to 0, and the regularization parameter is held fixed. Non-asymptotic convergence bounds to this limit in the $L_2$ sense are provided, together with upper bounds on the classification error that is shown to converge to the Bayes risk, therefore proving the Bayes-consistency of a variety of methods although the regularization term does not vanish. These results are particularly relevant to the one-class SVM, for which the regularization can not vanish by construction, and which is shown for the first time to be a consistent density level set estimator.

## 1 Introduction

Given $n$ i.i.d. copies $(X_1, Y_1), \ldots, (X_n, Y_n)$ of a random variable $(X, Y) \in \mathbb{R}^d \times \{-1, 1\}$, we study in this paper the limit and consistency of learning algorithms that solve the following problem:

$$\arg\min_{f \in \mathcal{H}_\sigma} \left\{ \frac{1}{n} \sum_{i=1}^{n} \phi\left(Y_i f(X_i)\right) + \lambda \| f \|_{\mathcal{H}_\sigma}^2 \right\} , \tag{1}$$

where $\phi : \mathbb{R} \to \mathbb{R}$ is a convex loss function and $\mathcal{H}_\sigma$ is the reproducing kernel Hilbert space (RKHS) of the normalized Gaussian radial basis function kernel (denoted simply Gaussian kernel below):

$$k_\sigma(x, x') = \frac{1}{\left(\sqrt{2\pi}\sigma\right)^d} \exp\left( \frac{-\| x - x' \|^2}{2\sigma^2} \right) , \;\; \sigma > 0 . \tag{2}$$

This framework encompasses in particular the classical support vector machine (SVM) [1] when $\phi(u) = \max(1 - u, 0)$. Recent years have witnessed important theoretical advances

aimed at understanding the behavior of such regularized algorithms when $n$ tends to infinity and $\lambda$ decreases to 0. In particular the consistency and convergence rates of the two-class SVM (see, e.g., [2, 3, 4] and references therein) have been studied in detail, as well as the shape of the asymptotic decision function [5, 6]. All results published so far study the case where $\lambda$ decreases as the number of points tends to infinity (or, equivalently, where $\lambda\sigma^{-d}$ converges to 0 if one uses the classical non-normalized version of the Gaussian kernel instead of (2)). Although it seems natural to reduce regularization as more and more training data are available –even more than natural, it is the spirit of regularization [7, 8]–, there is at least one important situation where $\lambda$ is typically held fixed: the one-class SVM [9]. In that case, the goal is to estimate an $\alpha$-quantile, that is, a subset of the input space $\mathcal{X}$ of given probability $\alpha$ with minimum volume. The estimation is performed by thresholding the function output by the one-class SVM, that is, the SVM (1) with only positive examples; in that case $\lambda$ is supposed to determine the quantile level[1]. Although it is known that the fraction of examples in the selected region converges to the desired quantile level $\alpha$ [9], it is still an open question whether the region converges towards a quantile, that is, a region of minimum volume. Besides, most theoretical results about the consistency and convergence rates of two-class SVM with vanishing regularization constant do not translate to the one-class case, as we are precisely in the seldom situation where the SVM is used with a regularization term that does not vanish as the sample size increases.

The main contribution of this paper is to show that Bayes consistency can be obtained for algorithms that solve (1) without decreasing $\lambda$, if instead the bandwidth $\sigma$ of the Gaussian kernel decreases at a suitable rate. We prove upper bounds on the convergence rate of the classification error towards the Bayes risk for a variety of functions $\phi$ and of distributions $P$, in particular for SVM (Theorem 6). Moreover, we provide an explicit description of the function asymptotically output by the algorithms, and establish converge rates towards this limit for the $L_2$ norm (Theorem 7). In particular, we show that the decision function output by the one-class SVM converges towards the density to be estimated, truncated at the level $2\lambda$ (Theorem 8); we finally show that this implies the consistency of one-class SVM as a density level estimator for the excess-mass functional [10] (Theorem 9).

Due to lack of space we limit ourselves in this extended abstract to the statement of the main results (Section 2) and sketch the proof of the main theorem (Theorem 3) that underlies all other results in Section 3. All detailed proofs are available in the companion paper [11].

## 2   Notations and main results

Let $(X, Y)$ be a pair of random variables taking values in $\mathbb{R}^d \times \{-1, 1\}$, with distribution $P$. We assume throughout this paper that the marginal distribution of $X$ is absolutely continuous with respect to Lebesgue measure with density $\rho : \mathbb{R}^d \to \mathbb{R}$, and that is has a support included in a compact set $\mathcal{X} \subset \mathbb{R}^d$. We denote $\eta : \mathbb{R}^d \to [0, 1]$ a measurable version of the conditional distribution of $Y = 1$ given $X$.

The normalized Gaussian radial basis function (RBF) kernel $k_\sigma$ with bandwidth parameter $\sigma > 0$ is defined for any $(x, x') \in \mathbb{R}^d \times \mathbb{R}^d$ by:

$$k_\sigma(x, x') = \frac{1}{\left(\sqrt{2\pi}\sigma\right)^d} \exp\left(\frac{-\| x - x' \|^2}{2\sigma^2}\right) \ ,$$

and the corresponding reproducing kernel Hilbert space (RKHS) is denoted by $\mathcal{H}_\sigma$. We note $\kappa_\sigma = \left(\sqrt{2\pi}\sigma\right)^{-d}$ the normalizing constant that ensures that the kernel integrates to 1.

Denoting by $\mathcal{M}$ the set of measurable real-valued functions on $\mathbb{R}^d$, we define several risks for functions $f \in \mathcal{M}$:

- The classification error rate, usually refered to as *(true) risk* of $f$, when $Y$ is predicted by the sign of $f(X)$, is denoted by

$$R(f) = P\left(\text{sign}\left(f(X)\right) \neq Y\right) .$$

- For a scalar $\lambda > 0$ fixed throughout this paper and a convex function $\phi : \mathbb{R} \to \mathbb{R}$, the $\phi$-risk regularized by the RKHS norm is defined, for any $\sigma > 0$ and $f \in \mathcal{H}_\sigma$, by

$$R_{\phi,\sigma}(f) = \mathbb{E}_P\left[\phi\left(Yf(X)\right)\right] + \lambda \| f \|^2_{\mathcal{H}_\sigma}$$

  Furthermore, for any real $r \geq 0$, we denote by $L(r)$ the Lipschitz constant of the restriction of $\phi$ to the interval $[-r, r]$. For example, for the hinge loss $\phi(u) = \max(0, 1-u)$ one can take $L(r) = 1$, and for the squared hinge loss $\phi(u) = \max(0, 1-u)^2$ one can take $L(r) = 2(r+1)$.

- Finally, the $L_2$-norm regularized $\phi$-risk is, for any $f \in \mathcal{M}$:

$$R_{\phi,0}(f) = \mathbb{E}_P\left[\phi\left(Yf(X)\right)\right] + \lambda \| f \|^2_{L_2}$$

  where,

$$\| f \|^2_{L_2} = \int_{\mathbb{R}^d} f(x)^2 dx \in [0, +\infty].$$

The minima of the three risk functionals defined above over their respective domains are denoted by $R^*$, $R^*_{\phi,\sigma}$ and $R^*_{\phi,0}$ respectively. Each of these risks has an empirical counterpart where the expectation with respect to $P$ is replaced by an average over an i.i.d. sample $T = \{(X_1, Y_1), \ldots, (X_n, Y_n)\}$. In particular, the following empirical version of $R_{\phi,\sigma}$ will be used

$$\forall \sigma > 0, f \in \mathcal{H}_\sigma, \quad \widehat{R}_{\phi,\sigma}(f) = \frac{1}{n} \sum_{i=1}^{n} \phi\left(Y_i f(X_i)\right) + \lambda \| f \|^2_{\mathcal{H}_\sigma} .$$

The main focus of this paper is the analysis of learning algorithms that minimize the empirical $\phi$-risk regularized by the RKHS norm $\widehat{R}_{\phi,\sigma}$, and their limit as the number of points tends to infinity and the kernel width $\sigma$ decreases to 0 at a suitable rate when $n$ tends to $\infty$, $\lambda$ being kept fixed. Roughly speaking, our main result shows that in this situation, if $\phi$ is a convex loss function, the minimization of $\widehat{R}_{\phi,\sigma}$ asymptotically amounts to minimizing $R_{\phi,0}$. This stems from the fact that the empirical average term in the definition of $\widehat{R}_{\phi,\sigma}$ converges to its corresponding expectation, while the norm in $\mathcal{H}_\sigma$ of a function $f$ decreases to its $L_2$ norm when $\sigma$ decreases to zero. To turn this intuition into a rigorous statement, we need a few more assumptions about the minimizer of $R_{\phi,0}$ and about $P$. First, we observe that the minimizer of $R_{\phi,0}$ is indeed well-defined and can often be explicitly computed:

**Lemma 1** *For any $x \in \mathbb{R}^d$, let*

$$f_{\phi,0}(x) = \underset{\alpha \in \mathbb{R}}{\arg\min} \left\{\rho(x)\left[\eta(x)\phi(\alpha) + (1-\eta)\phi(-\alpha)\right] + \lambda\alpha^2\right\} .$$

*Then $f_{\phi,0}$ is measurable and satisfies:*

$$R_{\phi,0}(f_{\phi,0}) = \inf_{f \in \mathcal{M}} R_{\phi,0}(f)$$

Second, we provide below a general result that shows how to control the excess $R_{\phi,0}$-risk of the empirical minimizer of the $R_{\phi,\sigma}$-risk, for which we need to recall the notion of modulus of continuity [12].

**Definition 2 (Modulus of continuity)** *Let $f$ be a Lebesgue measurable function from $\mathbb{R}^d$ to $\mathbb{R}$. Then its modulus of continuity in the $L_1$-norm is defined for any $\delta \geq 0$ as follows*

$$\omega(f, \delta) = \sup_{0 \leq \| t \| \leq \delta} \| f(. + t) - f(.) \|_{L_1} , \tag{3}$$

*where $\| t \|$ is the Euclidian norm of $t \in \mathbb{R}^d$.*

Our main result can now be stated as follows:

**Theorem 3 (Main Result)** *Let $\sigma_1 > \sigma > 0$, $0 < p < 2$, $\delta > 0$, and let $\hat{f}_{\phi,\sigma}$ denote a minimizer of the $\widehat{R}_{\phi,\sigma}$ risk over $\mathcal{H}_\sigma$. Assume that the marginal density $\rho$ is bounded, and let $M = \sup_{x \in \mathbb{R}^d} \rho(x)$. Then there exist constants $(K_i)_{i=1\ldots 4}$ (depending only on $p$, $\delta$, $\lambda$, $d$, and $M$) such that, for any $x > 0$, the following holds with probability greater than $1 - e^{-x}$ over the draw of the training data:*

$$
\begin{aligned}
R_{\phi,0}(\hat{f}_{\phi,\sigma}) - R^*_{\phi,0} \leq\ & K_1 L \left( \sqrt{\frac{\kappa_\sigma \phi(0)}{\lambda}} \right)^{\frac{4}{2+p}} \left( \frac{1}{\sigma} \right)^{\frac{[2+(2-p)(1+\delta)]d}{2+p}} \left( \frac{1}{n} \right)^{\frac{2}{2+p}} \\
& + K_2 L \left( \sqrt{\frac{\kappa_\sigma \phi(0)}{\lambda}} \right)^2 \left( \frac{1}{\sigma} \right)^d \frac{x}{n} \\
& + K_3 \frac{\sigma^2}{\sigma_1^2} \\
& + K_4 \omega(f_{\phi,0}, \sigma_1) .
\end{aligned}
\tag{4}
$$

The first two terms in the r.h.s. of (4) bound the estimation error associated with the gaussian RKHS, which naturally tends to be small when the number of training data increases and when the RKHS is 'small', i.e., when $\sigma$ is large. As is usually the case in such variance/bias splitings, the variance term here depends on the dimension $d$ of the input space. Note that it is also parametrized by both $p$ and $\delta$. The third term measures the error due to penalizing the $L_2$-norm of a fixed function in $\mathcal{H}_{\sigma_1}$ by its $\| . \|_{\mathcal{H}_\sigma}$-norm, with $0 < \sigma < \sigma_1$. This is a price to pay to get a small estimation error. As for the fourth term, it is a bound on the approximation error of the Gaussian RKHS. Note that, once $\lambda$ and $\sigma$ have been fixed, $\sigma_1$ remains a free variable parameterizing the bound itself.

In order to highlight the type of convergence rates one can obtain from Theorem 3, let us assume that the $\phi$ loss function is Lipschitz on $\mathbb{R}$ (e.g., take the hinge loss), and suppose that for some $0 \leq \beta \leq 1$, $c_1 > 0$, and for any $h \geq 0$, the function $f_{\phi,0}$ satisfies the following inequality

$$\omega(f_{\phi,0}, h) \leq c_1 h^\beta . \tag{5}$$

Then we can optimize the right hand side of (4) w.r.t. $\sigma_1$, $\sigma$, $p$ and $\delta$ by balancing the four terms. This eventually leads to:

$$R_{\phi,0}\left(\hat{f}_{\phi,\sigma}\right) - R^*_{\phi,0} = O_P \left( \left( \frac{1}{n} \right)^{\frac{2\beta}{4\beta + (2+\beta)d} - \epsilon} \right) , \tag{6}$$

for any $\epsilon > 0$. This rate is achieved by choosing

$$\sigma_1 = \left( \frac{1}{n} \right)^{\frac{2}{4\beta + (2+\beta)d} - \frac{\epsilon}{\beta}} , \tag{7}$$

$$\sigma = \sigma_1^{\frac{2+\beta}{2}} = \left( \frac{1}{n} \right)^{\frac{2+\beta}{4\beta + (2+\beta)d} - \frac{\epsilon(2+\beta)}{2\beta}} , \tag{8}$$

$p = 2$ and $\delta$ as small as possible (that is why an arbitray small quantity $\epsilon$ appears in the rate).

Theorem 3 shows that minimizing the $\widehat{R}_{\phi,\sigma}$ risk for well-chosen width $\sigma$ is a an algorithm consistant for the $R_{\phi,0}$-risk. In order to relate this consistency with more traditional measures of performance of learning algorithms, the next theorem shows that under a simple additionnal condition on $\phi$, $R_{\phi,0}$-risk-consistency implies Bayes consistency:

**Theorem 4** *If $\phi$ is convex, differentiable at $0$, with $\phi'(0) < 0$, then for every sequence of functions $(f_i)_{i \geq 1} \in \mathcal{M}$,*

$$\lim_{i \to +\infty} R_{\phi,0}(f_i) = R_{\phi,0}^* \implies \lim_{i \to +\infty} R(f_i) = R^*$$

This theorem results from a more general quantitative analysis of the relationship between the excess $R_{\phi,0}$-risk and the excess $R$-risk, in the spirit of [13]. In order to state a refined version in the particular case of the support vector machine algorithm, we first need the following definition:

**Definition 5** *We say that a distribution $P$ with $\rho$ as marginal density of $X$ w.r.t. Lebesgue measure has a low density exponent $\gamma \geq 0$ if there exists $(c_2, \epsilon_0) \in (0, +\infty)^2$ such that*

$$\forall \epsilon \in [0, \epsilon_0], \quad P\left(\left\{x \in \mathbb{R}^d : \rho(x) \leq \epsilon\right\}\right) \leq c_2 \epsilon^\gamma.$$

We are now in position to state a quantitative relationship between the excess $R_{\phi,0}$-risk and the excess $R$-risk in the case of support vector machines:

**Theorem 6** *Let $\phi_1(\alpha) = \max(1 - \alpha, 0)$ be the hinge loss function, and $\phi_2(\alpha) = \max(1 - \alpha, 0)^2$, be the squared hinge loss function. Then for any distribution $P$ with low density exponent $\gamma$, there exist constant $(K_1, K_2, r_1, r_2) \in (0, +\infty)^4$ such that for any $f \in \mathcal{M}$ with an excess $R_{\phi_1,0}$-risk upper bounded by $r_1$ the following holds:*

$$R(f) - R^* \leq K_1 \left(R_{\phi_1,0}(f) - R_{\phi_1,0}^*\right)^{\frac{\gamma}{2\gamma+1}},$$

*and if the excess regularized $R_{\phi_2,0}$-risk upper bounded by $r_2$ the following holds:*

$$R(f) - R^* \leq K_2 \left(R_{\phi_2,0}(f) - R_{\phi_2,0}^*\right)^{\frac{\gamma}{2\gamma+1}},$$

This result can be extended to any loss function through the introduction of variational arguments, in the spirit of [13]; we do not further explore this direction, but the reader is invited to consult [11] for more details. Hence we have proved the consistency of SVM, together with upper bounds on the convergence rates, in a situation where the effect of regularization does not vanish asymptotically.

Another consequence of the $R_{\phi,0}$-consistency of an algorithm is the $L_2$-convergence of the function output by the algorithm to the minimizer of the $R_{\phi,0}$-risk:

**Lemma 7** *For any $f \in \mathcal{M}$, the following holds:*

$$\| f - f_{\phi,0} \|_{L_2}^2 \leq \frac{1}{\lambda} \left(R_{\phi,0}(f) - R_{\phi,0}^*\right).$$

This result is particularly relevant to study algorithms whose objective are not binary classification. Consider for example the one-class SVM algorithm, which served as the initial motivation for this paper. Then we claim the following:

**Theorem 8** *Let $\rho_\lambda$ denote the density truncated as follows:*

$$\rho_\lambda(x) = \begin{cases} \frac{\rho(x)}{2\lambda} & \text{if } \rho(x) \leq 2\lambda, \\ 1 & \text{otherwise.} \end{cases} \tag{9}$$

*Let $\hat{f}_\sigma$ denote the function output by the one-class SVM, that is the function that solves (1) in the case $\phi$ is the hinge-loss function and $Y_i = 1$ for all $i \in \{1, \ldots, n\}$. Then, under the general conditions of Theorem 3, for $\sigma$ choosen as in Equation (8),*

$$\lim_{n \to +\infty} \| \hat{f}_\sigma - \rho_\lambda \|_{L_2} = 0 \; .$$

An interesting by-product of this theorem is the consistency of the one-class SVM algorithm for density level set estimation:

**Theorem 9** *Let $0 < \mu < 2\lambda < M$, let $C_\mu$ be the level set of the density function $\rho$ at level $\mu$, and $\widehat{C}_\mu$ be the level set of $2\lambda\hat{f}_\sigma$ at level $\mu$, where $\hat{f}_\sigma$ is still the function outptut by the one-class SVM. For any distribution $Q$, for any subset $C$ of $\mathbb{R}^d$, define the excess-mass of $C$ with respect to $Q$ as follows:*

$$H_Q(C) = Q(C) - \mu Leb(C) \; , \tag{10}$$

*where Leb is the Lebesgue measure. Then, under the general assumptions of Theorem 3, we have*

$$\lim_{n \to +\infty} H_P(C_\mu) - H_P\left(\widehat{C}_\mu\right) = 0 \; , \tag{11}$$

*for $\sigma$ choosen as in Equation (8).*

The excess-mass functional was first introduced in [10] to assess the quality of density level set estimators. It is maximized by the true density level set $C_\mu$ and acts as a risk functional in the one-class framework. The proof ef Theorem 9 is based on the following result: if $\hat{\rho}$ is a density estimator converging to the true density $\rho$ in the $L_2$ sense, then for any fixed $0 < \mu < \sup \{\rho\}$, the excess mass of the level set of $\hat{\rho}$ at level $\mu$ converges to the excess mass of $C_\mu$. In other words, as is the case in the classification framework, plug-in estimators built on $L_2$-consistent density estimators are consistent with respect to the excess mass.

## 3   Proof of Theorem 3 (sketch)

In this section we sketch the proof of the main learning theorem of this contribution, which underlies most other results stated in Section 2 The proof of Theorem 3 is based on the following decomposition of the excess $R_{\phi,0}$-risk for the minimizer $\hat{f}_{\phi,\sigma}$ of $\widehat{R}_{\phi,\sigma}$, valid for any $0 < \sigma < \sqrt{2}\sigma_1$ and any sample $(x_i, y_i)_{i=1,\ldots,n}$:

$$\begin{aligned} R_{\phi,0}(\hat{f}_{\phi,\sigma}) - R_{\phi,0}^* &= \left[ R_{\phi,0}\left(\hat{f}_{\phi,\sigma}\right) - R_{\phi,\sigma}\left(\hat{f}_{\phi,\sigma}\right) \right] \\ &+ \left[ R_{\phi,\sigma}(\hat{f}_{\phi,\sigma}) - R_{\phi,\sigma}^* \right] \\ &+ \left[ R_{\phi,\sigma}^* - R_{\phi,\sigma}(k_{\sigma_1} * f_{\phi,0}) \right] \\ &+ \left[ R_{\phi,\sigma}(k_{\sigma_1} * f_{\phi,0}) - R_{\phi,0}(k_{\sigma_1} * f_{\phi,0}) \right] \\ &+ \left[ R_{\phi,0}(k_{\sigma_1} * f_{\phi,0}) - R_{\phi,0}^* \right] \; . \end{aligned} \tag{12}$$

It can be shown that $k_{\sigma_1} * f_{\phi,0} \in \mathcal{H}_{\sqrt{2}\sigma_1} \subset \mathcal{H}_\sigma \subset L_{2(\mathbb{R}^d)}$ which justifies the introduction of $R_{\phi,\sigma}(k_{\sigma_1} * f_{\phi,0})$ and $R_{\phi,0}(k_{\sigma_1} * f_{\phi,0})$. By studying the relationship between the Gaussian RKHS norm and the $L_2$ norm, it can be shown that

$$R_{\phi,0}\left(\hat{f}_{\phi,\sigma}\right) - R_{\phi,\sigma}\left(\hat{f}_{\phi,\sigma}\right) = \lambda \left( \| \hat{f}_{\phi,\sigma} \|_{L_2}^2 - \| \hat{f}_{\phi,\sigma} \|_{\mathcal{H}_\sigma}^2 \right) \leq 0,$$

while the following stems from the definition of $R_{\phi,\sigma}^*$:

$$R_{\phi,\sigma}^* - R_{\phi,\sigma}(k_{\sigma_1} * f_{\phi,0}) \leq 0.$$

Hence, controlling $R_{\phi,0}(\hat{f}_{\phi,\sigma}) - R_{\phi,0}^*$ boils down to controlling each of the remaining three terms in (12).

- The second term in (12) is usually referred to as the sample error or estimation error. The control of such quantities has been the topic of much research recently, including for example [14, 15, 16, 17, 18, 4]. Using estimates of local Rademacher complexities through covering numbers for the Gaussian RKHS due to [4], the following result can be shown:

  **Lemma 10** *For any $\sigma > 0$ small enough, let $\hat{f}_{\phi,\sigma}$ be the minimizer of the $\widehat{R}_{\phi,\sigma}$-risk on a sample of size $n$, where $\phi$ is a convex loss function. For any $0 < p < 2$, $\delta > 0$, and $x \geq 1$, the following holds with probability at least $1 - e^x$ over the draw of the sample:*

  $$R_{\phi,\sigma}(\hat{f}_{\phi,\sigma}) - R_{\phi,\sigma}(f_{\phi,\sigma}) \leq K_1 L \left( \sqrt{\frac{\kappa_\sigma \phi(0)}{\lambda}} \right)^{\frac{4}{2+p}} \left( \frac{1}{\sigma} \right)^{\frac{[2+(2-p)(1+\delta)]d}{2+p}} \left( \frac{1}{n} \right)^{\frac{2}{2+p}}$$

  $$+ K_2 L \left( \sqrt{\frac{\kappa_\sigma \phi(0)}{\lambda}} \right)^2 \left( \frac{1}{\sigma} \right)^d \frac{x}{n} ,$$

  *where $K_1$ and $K_2$ are positive constants depending neither on $\sigma$, nor on $n$.*

- In order to upper bound the fourth term in (12), the analysis of the convergence of the Gaussian RKHS norm towards the $L_2$ norm when the bandwidth of the kernel tends to 0 leads to:

  $$R_{\phi,\sigma}(k_{\sigma_1} * f_{\phi,0}) - R_{\phi,0}(k_{\sigma_1} * f_{\phi,0}) = \| k_{\sigma_1} * f_{\phi,0} \|_{\mathcal{H}_\sigma}^2 - \| k_{\sigma_1} * f_{\phi,0} \|_{L_2}^2$$

  $$\leq \frac{\sigma^2}{2\sigma_1^2} \| f_{\phi,0} \|_{L_2}^2$$

  $$\leq \frac{\phi(0)\sigma^2}{2\lambda\sigma_1^2} .$$

- The fifth term in (12) corresponds to the approximation error. It can be shown that for any bounded function in $L_1(\mathbb{R}^d)$ and all $\sigma > 0$, the following holds:

  $$\| k_\sigma * f - f \|_{L_1} \leq (1 + \sqrt{d})\omega(f,\sigma) , \tag{13}$$

  where $\omega(f,.)$ denotes the modulus of continuity of $f$ in the $L_1$ norm. From this the following inequality can be derived:

  $$R_{\phi,0}(k_{\sigma_1} * f_{\phi,0}) - R_{\phi,0}(f_{\phi,0})$$

  $$\leq (2\lambda \| f_{\phi,0} \|_{L_\infty} + L(\| f_{\phi,0} \|_{L_\infty}) M) \left( 1 + \sqrt{d} \right) \omega(f_{\phi,0}, \sigma_1) .$$

## 4  Conclusion

We have shown that consistency of learning algorithms that minimize a regularized empirical risk can be obtained even when the so-called regularization term does not asymptotically vanish, and derived the consistency of one-class SVM as a density level set estimator. Our method of proof is based on an unusual decomposition of the excess risk due to the presence of the regularization term, which plays an important role in the determination of the asymptotic limit of the function that minimizes the empirical risk. Although the upper bounds on the convergence rates we obtain are not optimal, they provide a first step toward the analysis of learning algorithms in this context.

**Acknowledgments**

The authors are grateful to Stéphane Boucheron, Pascal Massart and Ingo Steinwart for fruitful discussions. This work was supported by the ACI "Nouvelles interfaces des Mathématiques" of the French Ministry for Research, and by the IST Program of the European Community, under the Pascal Network of Excellence, IST-2002-506778.

## Footnotes

[1] While the original formulation of the one-class SVM involves a parameter $\nu$, there is asymptotically a one-to-one correspondance between $\lambda$ and $\nu$

# References

[1] B. E. Boser, I. M. Guyon, and V. N. Vapnik. A training algorithm for optimal margin classifiers. In *Proceedings of the 5th annual ACM workshop on Computational Learning Theory*, pages 144–152. ACM Press, 1992.

[2] I. Steinwart. Support vector machines are universally consistent. *J. Complexity*, 18:768–791, 2002.

[3] T. Zhang. Statistical behavior and consistency of classification methods based on convex risk minimization. *Ann. Stat.*, 32:56–134, 2004.

[4] I. Steinwart and C. Scovel. Fast rates for support vector machines using gaussian kernels. Technical report, Los Alamos National Laboratory, 2004. submitted to Annals of Statistics.

[5] I. Steinwart. Sparseness of support vector machines. *J. Mach. Learn. Res.*, 4:1071–1105, 2003.

[6] P. L. Bartlett and A. Tewari. Sparseness vs estimating conditional probabilities: Some asymptotic results. In *Lecture Notes in Computer Science*, volume 3120, pages 564–578. Springer, 2004.

[7] A.N. Tikhonov and V.Y. Arsenin. *Solutions of ill-posed problems*. W.H. Winston, Washington, D.C., 1977.

[8] B. W. Silverman. On the estimation of a probability density function by the maximum penalized likelihood method. *Ann. Stat.*, 10:795–810, 1982.

[9] B. Schölkopf, J. C. Platt, J. Shawe-Taylor, A. J. Smola, and R. C. Williamson. Estimating the support of a high-dimensional distribution. *Neural Comput.*, 13:1443–1471, 2001.

[10] J. A. Hartigan. Estimation of a convex density contour in two dimensions. *J. Amer. Statist. Assoc.*, 82(397):267–270, 1987.

[11] R. Vert and J.-P. Vert. Consistency and convergence rates of one-class svm and related algorithms. *J. Mach. Learn. Res.*, 2006. To appear.

[12] R. A. DeVore and G. G. Lorentz. *Constructive Approximation*. Springer Grundlehren der Mathematischen Wissenschaften. Springer Verlag, 1993.

[13] P.I. Bartlett, M.I. Jordan, and J.D. McAuliffe. Convexity, classification and risk bounds. Technical Report 638, UC Berkeley Statistics, 2003.

[14] A. B. Tsybakov. On nonparametric estimation of density level sets. *Ann. Stat.*, 25:948–969, June 1997.

[15] E. Mammen and A. Tsybakov. Smooth discrimination analysis. *Ann. Stat.*, 27(6):1808–1829, 1999.

[16] P. Massart. Some applications of concentration inequalities to statistics. *Ann. Fac. Sc. Toulouse*, IX(2):245–303, 2000.

[17] P. L. Bartlett, O. Bousquet, and S. Mendelson. Local rademacher complexities. *Annals of Statistics*, 2005. To appear.

[18] V. Koltchinskii. Localized rademacher complexities. Manuscript, september 2003.
